# Toward Provably Correct Feature Selection in Arbitrary Domains

**Dimitris Margaritis**
Department of Computer Science
Iowa State University
Ames, IA 50010, USA
`dmarg@cs.iastate.edu`

## Abstract

In this paper we address the problem of provably correct feature selection in arbitrary domains. An optimal solution to the problem is a Markov boundary, which is a minimal set of features that make the probability distribution of a target variable conditionally invariant to the state of all other features in the domain. While numerous algorithms for this problem have been proposed, their theoretical correctness and practical behavior under arbitrary probability distributions is unclear. We address this by introducing the Markov Boundary Theorem that precisely characterizes the properties of an ideal Markov boundary, and use it to develop algorithms that learn a more general boundary that can capture complex interactions that only appear when the values of multiple features are considered together. We introduce two algorithms: an exact, provably correct one as well a more practical randomized anytime version, and show that they perform well on artificial as well as benchmark and real-world data sets. Throughout the paper we make minimal assumptions that consist of only a general set of axioms that hold for every probability distribution, which gives these algorithms universal applicability.

## 1  Introduction and Motivation

The problem of feature selection has a long history due to its significance in a wide range of important problems, from early ones like pattern recognition to recent ones such as text categorization, gene expression analysis and others. In such domains, using all available features may be prohibitively expensive, unnecessarily wasteful, and may lead to poor generalization performance, especially in the presence of irrelevant or redundant features. Thus, selecting a subset of features of the domain for use in subsequent application of machine learning algorithms has become a standard preprocessing step. A typical task of these algorithms is *learning a classifier*: Given a number of input features and a quantity of interest, called the *target variable*, choose a member of a family of classifiers that can predict the target variable's value as well as possible. Another task is *understanding* the domain and the quantities that interact with the target quantity.

Many algorithms have been proposed for feature selection. Unfortunately, little attention has been paid to the issue of their behavior under a variety of application domains that can be encountered in practice. In particular, it is known that many can fail under certain probability distributions such as ones that contain a (near) parity function [1], which contain interactions that only appear when the values of multiple features are considered together. There is therefore an acute need for algorithms that are widely applicable and can be theoretically proven to work under any probability distribution. In this paper we present two such algorithms, an exact and a more practical randomized approximate one. We use the observation (first made in Koller and Sahami [2]) that an optimal solution to the problem is a Markov boundary, defined to be a minimal set of features that make the probability distribution of a target variable conditionally invariant to the state of all other features in the domain (a more precise definition is given later in Section 3) and present a family of algorithms for learning

the Markov boundary of a target variable in arbitrary domains. We first introduce a theorem that exactly characterizes the minimal set of features necessary for probabilistically isolating a variable, and then relax this definition to derive a family of algorithms that learn a parameterized approximation of the ideal boundary that are *provably correct* under a minimal set of assumptions, including a set of axioms that hold for any probability distribution.

In the following section we present work on feature selection, followed by notation and definitions in Section 3. We subsequently introduce an important theorem and the aforementioned parameterized family of algorithms in Sections 4 and 5 respectively, including a practical anytime version. We evaluate these algorithms in Section 6 and conclude in Section 7.

## 2 Related Work

Numerous algorithms have been proposed for feature selection. At the highest level algorithms can be classified as *filter*, *wrapper*, or *embedded* methods. Filter methods work without consulting the classifier (if any) that will make use of their output i.e., the resulting set of selected features. They therefore have typically wider applicability since they are not tied to any particular classifier family. In contrast, wrappers make the classifier an integral part of their operation, repeatedly invoking it to evaluate each of a sequence of feature subsets, and selecting the subset that results in minimum estimated classification error (for that particular classifier). Finally, embedded algorithms are classifier-learning algorithms that perform feature selection implicitly during their operation e.g., decision tree learners.

Early work was motivated by the problem of pattern recognition which inherently contains a large number of features (pixels, regions, signal responses at multiple frequencies etc.). Narendra and Fukunaga [3] first cast feature selection as a problem of maximization of an objective function over the set of features to use, and proposed a number of search approaches including *forward selection* and *backward elimination*. Later work by machine learning researchers includes the FOCUS algorithm of Almuallim and Dietterich [4], which is a filter method for deterministic, noise-free domains. The RELIEF algorithm [5] instead uses a randomized selection of data points to update a weight assigned to each feature, selecting the features whose weight exceeds a given threshold. A large number of additional algorithms have appeared in the literature, too many to list here—good surveys are included in Dash and Liu [6]; Guyon and Elisseeff [1]; Liu and Motoda [7]. An important concept for feature subset selection is relevance. Several notions of relevance are discussed in a number of important papers such as Blum and Langley [8]; Kohavi and John [9]. The argument that the problem of feature selection can be cast as the problem of Markov blanket discovery was first made convincingly in Koller and Sahami [2], who also presented an algorithm for learning an approximate Markov blanket using mutual information. Other algorithms include the GS algorithm [10], originally developed for learning of the structure of a Bayesian network of a domain, and extensions to it [11] including the recent MMMB algorithm [12]. Meinshausen and Bühlmann [13] recently proposed an optimal theoretical solution to the problem of learning the neighborhood of a Markov network when the distribution of the domain can be assumed to be a multidimensional Gaussian i.e., linear relations among features with Gaussian noise. This assumption implies that the Composition axiom holds in the domain (see Pearl [14] for a definition of Composition); the difference with our work is that we address here the problem in general domains where it may not necessarily hold.

## 3 Notation and Preliminaries

In this section we present notation, fundamental definitions and axioms that will be subsequently used in the rest of the paper. We use the term "feature" and "variable" interchangeably, and denote variables by capital letters ($X$, $Y$ etc.) and sets of variables by bold letters ($\mathbf{S}$, $\mathbf{T}$ etc.). We denote the set of all variables/features in the domain (the "universe") by $\mathcal{U}$. All algorithms presented are *independence-based*, learning the Markov boundary of a given target variable using the truth value of a number of conditional independence statements. The use of conditional independence for feature selection subsumes many other criteria proposed in the literature. In particular, the use of classification accuracy of the target variable can be seen as a special case of testing for its conditional independence with some of its predictor variables (conditional on the subset selected at any given moment). A benefit of using conditional independence is that, while classification error estimates depend on the classifier family used, conditional independence does not. In addition, algorithms utilizing conditional independence for feature selection are applicable to all domain types,

e.g., discrete, ordinal, continuous with non-linear or arbitrary non-degenerate associations or mixed domains, as long as a reliable estimate of probabilistic independence is available.

We denote probabilistic independence by the symbol " $\perp\!\!\!\perp$ " i.e., $(\mathbf{X} \perp\!\!\!\perp \mathbf{Y} \mid \mathbf{Z})$ denotes the fact that the variables in set $\mathbf{X}$ are (jointly) conditionally independent from those in set $\mathbf{Y}$ given the values of the variables in set $\mathbf{Z}$; $(\mathbf{X} \not\perp\!\!\!\perp \mathbf{Y} \mid \mathbf{Z})$ denotes their conditional dependence. We assume the existence of a *probabilistic independence query oracle* that is available to answer any query of the form $(\mathbf{X}, \mathbf{Y} \mid \mathbf{Z})$, corresponding to the question "Is the set of variables in $\mathbf{X}$ independent of the variables in $\mathbf{Y}$ given the value of the variables in $\mathbf{Z}$?" (This is similar to the approach of learning from statistical queries of Kearns and Vazirani [15].) In practice however, such an oracle does not exist, but can be approximated by a statistical independence test on a data set. Many tests of independence have appeared and studied extensively in the statistical literature over the last century; in this work we use the $\chi^2$ (chi-square) test of independence [16].

A Markov blanket of variable $X$ is a set of variables such that, after fixing (by "knowing") the value of all of its members, the set of remaining variables in the domain, taken together as a single set-valued variable, are statistically independent of $X$. More precisely, we have the following definition.

**Definition 1.** *A set of variables* $\mathbf{S} \subseteq \mathcal{U}$ *is called a **Markov blanket** of variable* $X$ *if and only if* $(X \perp\!\!\!\perp \mathcal{U} - \mathbf{S} - \{X\} \mid \mathbf{S})$.

Intuitively, a Markov blanket $\mathbf{S}$ of $X$ captures all the information in the remaining domain variables $\mathcal{U} - \mathbf{S} - \{X\}$ that can affect the probability distribution of $X$, making their value redundant as far as $X$ is concerned (given $\mathbf{S}$). The blanket therefore captures the essence of the feature selection problem for target variable $X$: By completely "shielding" $X$, a Markov blanket precludes the existence of any possible information about $X$ that can come from variables not in the blanket, making it an ideal solution to the feature selection problem. A minimal Markov blanket is called a Markov boundary.

**Definition 2.** *A set of variables* $\mathbf{S} \subseteq \mathcal{U} - \{X\}$ *is called a **Markov boundary** of variable* $X$ *if it is a minimal Markov blanket of* $X$ *i.e., none of its proper subsets is a Markov blanket.*

Pearl [14] proved that that the axioms of Symmetry, Decomposition, Weak Union, and Intersection are sufficient to guarantee a unique Markov boundary. These are shown below together with the axiom of Contraction.

$$
\begin{array}{rl}
\textbf{(Symmetry)} & (\mathbf{X} \perp\!\!\!\perp \mathbf{Y} \mid \mathbf{Z}) \implies (\mathbf{Y} \perp\!\!\!\perp \mathbf{X} \mid \mathbf{Z}) \\
\textbf{(Decomposition)} & (\mathbf{X} \perp\!\!\!\perp \mathbf{Y} \cup \mathbf{W} \mid \mathbf{Z}) \implies (\mathbf{X} \perp\!\!\!\perp \mathbf{Y} \mid \mathbf{Z}) \wedge (\mathbf{X} \perp\!\!\!\perp \mathbf{W} \mid \mathbf{Z}) \\
\textbf{(Weak Union)} & (\mathbf{X} \perp\!\!\!\perp \mathbf{Y} \cup \mathbf{W} \mid \mathbf{Z}) \implies (\mathbf{X} \perp\!\!\!\perp \mathbf{Y} \mid \mathbf{Z} \cup \mathbf{W}) \\
\textbf{(Contraction)} & (\mathbf{X} \perp\!\!\!\perp \mathbf{Y} \mid \mathbf{Z}) \wedge (\mathbf{X} \perp\!\!\!\perp \mathbf{W} \mid \mathbf{Y} \cup \mathbf{Z}) \implies (\mathbf{X} \perp\!\!\!\perp \mathbf{Y} \cup \mathbf{W} \mid \mathbf{Z}) \\
\textbf{(Intersection)} & (\mathbf{X} \perp\!\!\!\perp \mathbf{Y} \mid \mathbf{Z} \cup \mathbf{W}) \wedge (\mathbf{X} \perp\!\!\!\perp \mathbf{W} \mid \mathbf{Z} \cup \mathbf{Y}) \implies (\mathbf{X} \perp\!\!\!\perp \mathbf{Y} \cup \mathbf{W} \mid \mathbf{Z})
\end{array}
\tag{1}
$$

The Symmetry, Decomposition, Contraction and Weak Union axioms are very general: they are *necessary* axioms for the probabilistic definition of independence i.e., they hold in *every* probability distribution, as their proofs are based on the axioms of probability theory. Intersection is not universal but it holds in distributions that are positive, i.e., any value combination of the domain variables has a non-zero probability of occurring.

## 4 The Markov Boundary Theorem

According to Definition 2, a Markov boundary is a minimal Markov blanket. We first introduce a theorem that provides an alternative, equivalent definition of the concept of Markov boundary that we will relax later in the paper to produce a more general boundary definition.

**Theorem 1 (Markov Boundary Theorem).** *Assuming that the Decomposition and Contraction axioms hold,* $\mathbf{S} \subseteq \mathcal{U} - \{X\}$ *is a Markov boundary of variable* $X \in \mathcal{U}$ *if and only if*

$$
\forall \mathbf{T} \subseteq \mathcal{U} - \{X\}, \left\{ \mathbf{T} \subseteq \mathcal{U} - \mathbf{S} \iff (X \perp\!\!\!\perp \mathbf{T} \mid \mathbf{S} - \mathbf{T}) \right\}.
\tag{2}
$$

A detailed proof cannot be included here due to space constraints but a proof sketch appears in Appendix A. According to the above theorem, a Markov boundary $\mathbf{S}$ partitions the powerset of $\mathcal{U} - \{X\}$ into two parts: (a) set $\mathcal{P}_1$ that contains all subsets of $\mathcal{U} - \mathbf{S}$, and (b) set $\mathcal{P}_2$ containing the remaining subsets. All sets in $\mathcal{P}_1$ are conditionally independent of $X$, and all sets in $\mathcal{P}_2$ are conditionally dependent with $X$.

Intuitively, the two directions of the logical equivalence relation of Eq. (2) correspond to the concept of Markov blanket and its minimality i.e., the equation

$$
\forall \mathbf{T} \subseteq \mathcal{U} - \{X\}, \left\{ \mathbf{T} \subseteq \mathcal{U} - \mathbf{S} \implies (X \perp\!\!\!\perp \mathbf{T} \mid \mathbf{S} - \mathbf{T}) \right\}
$$

**Algorithm 1** The abstract $\mathrm{GS}^{(m)}(X)$ algorithm. Returns an $m$-Markov boundary of $X$.

```
 1: S ← ∅
 2: /* Growing phase. */
 3: for all Y ⊆ U − S − {X} such that 1 ≤ |Y| ≤ m do
 4:     if (X ⊥̸ Y | S) then
 5:         S ← S ∪ Y
 6:         goto line 3            /* Restart loop. */
 7: /* Shrinking phase. */
 8: for all Y ∈ S do
 9:     if (X ⊥⊥ Y | S − {Y}) then
10:         S ← S − {Y}
11:         goto line 8            /* Restart loop. */
12: return S
```

or, equivalently, $(\forall\, \mathbf{T} \subseteq \mathcal{U} - \mathbf{S} - \{X\},\ (X \perp\!\!\!\perp \mathbf{T} \mid \mathbf{S}))$ (as $\mathbf{T}$ and $\mathbf{S}$ are disjoint) corresponds to the definition of Markov blanket, as it includes $\mathbf{T} = \mathcal{U} - \mathbf{S} - \{X\}$. In the opposite direction, the contrapositive form is

$$\forall\, \mathbf{T} \subseteq \mathcal{U} - \{X\}, \Big\{ \mathbf{T} \not\subseteq \mathcal{U} - \mathbf{S} \implies (X \perp\!\!\!\!\not\perp \mathbf{T} \mid \mathbf{S} - \mathbf{T}) \Big\}.$$

This corresponds to the concept of minimality of the Markov boundary: It states that all sets that contain a part of $\mathbf{S}$ cannot be independent of $X$ given the remainder of $\mathbf{S}$. Informally, this is because if there existed some set $\mathbf{T}$ that contained a non-empty subset $\mathbf{T}'$ of $\mathbf{S}$ such that $(X \perp\!\!\!\perp \mathbf{T} \mid \mathbf{S} - \mathbf{T})$, then one would be able to shrink $\mathbf{S}$ by $\mathbf{T}'$ (by the property of Contraction) and therefore $\mathbf{S}$ would not be minimal (more details in Appendix A).

## 5  A Family of Algorithms for Arbitrary Domains

Theorem 1 defines conditions that precisely characterize a Markov boundary and thus can be thought of as an alternative definition of a boundary. By relaxing these conditions we can produce a more general definition. In particular, an $m$-Markov boundary is defined as follows.

**Definition 3.** *A set of variables* $\mathbf{S} \subseteq \mathcal{U} - \{X\}$ *of a domain* $\mathcal{U}$ *is called an* $m$-***Markov boundary*** *of variable* $X \in \mathcal{U}$ *if and only if*

$$\forall\, \mathbf{T} \subseteq \mathcal{U} - \{X\} \text{ such that } |\mathbf{T}| \leq m, \Big\{ \mathbf{T} \subseteq \mathcal{U} - \mathbf{S} \iff (X \perp\!\!\!\perp \mathbf{T} \mid \mathbf{S} - \mathbf{T}) \Big\}.$$

We call the parameter $m$ of an $m$-Markov boundary the *Markov boundary margin*. Intuitively, an $m$-boundary $\mathbf{S}$ guarantees that (a) all subsets of its complement (excluding $X$) of size $m$ or smaller are independent of $X$ given $\mathbf{S}$, and (b) all sets $\mathbf{T}$ of size $m$ or smaller that are not subsets of its complement are dependent of $X$ given the part of $\mathbf{S}$ that is not contained in $\mathbf{T}$. This definition is a special case of the properties of a boundary stated in Theorem 1, with each set $\mathbf{T}$ mentioned in the theorem now restricted to having size $m$ or smaller. For $m = n - 1$, where $n = |\mathcal{U}|$, the condition $|\mathbf{T}| \leq m$ is always satisfied and can be omitted; in this case the definition of an $(n - 1)$-Markov boundary results in exactly Eq. (2) of Theorem 1.

We now present an algorithm called $\mathrm{GS}^{(m)}$, shown in Algorithm 1, that provably correctly learns an $m$-boundary of a target variable $X$. $\mathrm{GS}^{(m)}$ operates in two phases, a *growing* and a *shrinking* phase (hence the acronym). During the growing phase it examines sets of variables of size up to $m$, where $m$ is a user-specified parameter. During the shrinking phase, *single* variables are examined for conditional independence and possible removal from $\mathbf{S}$ (examining sets in the shrinking phase is *not* necessary for provably correct operation—see Appendix B). The orders of examination of the sets for possible addition and deletion from the candidate boundary are left intentionally unspecified in Algorithm 1—one can therefore view it as an abstract representative of a family of algorithms, with each member specifying one such ordering. All members of this family are $m$-correct, as the proof of correctness does not depend on the ordering. In practice numerous choices for the ordering exist; one possibility is to examine the sets in the growing phase in order of increasing set size and, for each such size, in order of decreasing conditional mutual information $I(X, \mathbf{Y}, \mathbf{S})$ between $X$ and $\mathbf{Y}$ given $\mathbf{S}$. The rationale for this heuristic choice is that (usually) tests with smaller conditional sets tend to be more reliable, and sorting by mutual information tends to lessen the chance of adding false members of the Markov boundary. We used this implementation in all our experiments, presented later in Section 6.

Intuitively, the margin represents a trade-off between sample and computational complexity and completeness: For $m = n - 1 = |\mathcal{U}| - 1$, the algorithm returns a Markov boundary in unrestricted

**Algorithm 2** The $\text{RGS}^{(m,k)}(X)$ algorithm, a randomized anytime version of the $\text{GS}^{(m)}$ algorithm, utilizing $k$ random subsets for the growing phase.

```
 1: S ← ∅
 2: /* Growing phase. */
 3: repeat
 4:     Schanged ← false
 5:     Y ← subset of 𝒰 − S − {X} of size 1 ≤ |Y| ≤ m of maximum dependence out of k random subsets
 6:     if (X ⊥̸ Y | S) then
 7:         S ← S ∪ Y
 8:         Schanged ← true
 9: until Schanged = false
10: /* Shrinking phase. */
11: for all Y ∈ S do
12:     if (X ⊥ Y | S − {Y}) then
13:         S ← S − {Y}
14:         goto line 11          /* Restart loop. */
15: return S
```

(arbitrary) domains. For $1 \leq m < n-1$, $\text{GS}^{(m)}$ may recover the correct boundary depending on characteristics of the domain. For example, it will recover the correct boundary in domains containing embedded parity functions such that the number of variables involved in every $k$-bit parity function is $m+1$ or less i.e., if $k \leq m+1$ (parity functions are corner cases in the space of probability distributions that are known to be hard to learn [17]). The proof of $m$-correctness of $\text{GS}^{(m)}$ is included in Appendix B. Note that it is based on Theorem 1 and the universal axioms of Eqs. (1) only i.e., Intersection is not needed, and thus it is widely applicable (to any domain).

**A Practical Randomized Anytime Version**

While $\text{GS}^{(m)}$ is provably correct even in difficult domains such as those that contain parity functions, it may be impractical with a large number of features as its asymptotic complexity is $O(n^m)$. We therefore also we here provide a more practical randomized version called $\text{RGS}^{(m,k)}$ (Randomized $\text{GS}^{(m)}$), shown in Algorithm 2. The $\text{RGS}^{(m,k)}$ algorithm has an additional parameter $k$ that limits its computational requirements: instead of exhaustively examining all possible subsets of $(\mathcal{U}-S-\{X\})$ (as $\text{GS}^{(m)}$ does), it instead samples $k$ subsets from the set of all possible subsets of $(\mathcal{U} - S - \{X\})$, where $k$ is user-specified. It is therefore a randomized algorithm that becomes equivalent to $\text{GS}^{(m)}$ given a large enough $k$. Many possibilities for the method of random selection of the subsets exist; in our experiments we select a subset $Y = \{Y_i\}$ ($1 \leq |Y| \leq m$) with probability proportional to $\sum_{i=1}^{|Y|}(1/p(X, Y_i \mid S))$, where $p(X, Y_i \mid S)$ is the p-value of the corresponding (univariate) test between $X$ and $Y_i$ given $S$, which has a low computational cost.

The $\text{RGS}^{(m,k)}$ algorithm is useful in situations where the amount of time to produce an answer may be limited and/or the limit unknown beforehand: it is easy to show that the growing phase of $\text{GS}^{(m)}$ produces an an upper-bound of the $m$-boundary of $X$. Therefore, the $\text{RGS}^{(m,k)}$ algorithm, if interrupted, will return an approximation of this upper bound. Moreover, if there exists time for the shrinking phase to be executed (which conducts a number of tests linear in $n$ and is thus fast), extraneous variables will be removed and a minimal blanket (boundary) approximation will be returned. These features make it an *anytime* algorithm, which is a more appropriate choice for situations where critical events may occur that require the interruption of computation, e.g., during the planning phase of a robot, which may be interrupted at any time due to an urgent external event that requires a decision to be made based on the present state's feature values.

## 6  Experiments

We evaluated the $\text{GS}^{(m)}$ and the $\text{RGS}^{(m,k)}$ algorithms on synthetic as well as real-world and benchmark data sets. We first systematically examined the performance on the task of recovering near-parity functions, which are known to be hard to learn [17]. We compared $\text{GS}^{(m)}$ and $\text{RGS}^{(m,k)}$ with respect to accuracy of recovery of the original boundary as well as computational cost. We generated domains of sizes ranging from 10 to 100 variables, of which 4 variables ($X_1$ to $X_4$) were related through a near-parity relation with bit probability 0.60 and various degrees of noise. The remaining independent variables ($X_5$ to $X_n$) act as "dis-

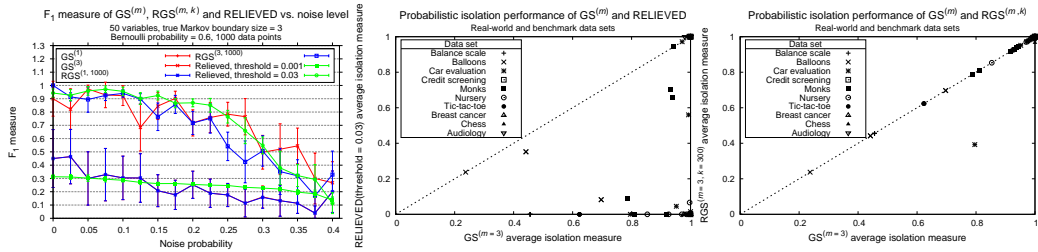

Figure 2: **Left:** $F_1$ measure of $GS^{(m)}$, $RGS^{(m,k)}$ and RELIEVED under increasing amounts of noise. **Middle:** Probabilistic isolation performance comparison between $GS^{(3)}$ and RELIEVED on real-world and benchmark data sets. **Right:** Same for $GS^{(3)}$ and $RGS^{(3,1000)}$.

tractors" and had randomly assigned probabilities i.e., the correct boundary of $X_1$ is $\mathbf{B}_1 = \{X_2, X_3, X_4\}$. In such domains, learning the boundary of $X_1$ is difficult because of the large number of distractors and because each $X_i \in \mathbf{B}_1$ is independent of $X_1$ given any proper subset of $\mathbf{B}_1 - \{X_i\}$ (they only become dependent when including all of them in the conditioning set).

To measure an algorithm's feature selection performance, accuracy (fraction of variables correctly included or excluded) is inappropriate as the accuracy of trivial algorithms such as returning the empty set will tend to 1 as $n$ increases. Precision and recall are therefore more appropriate, with precision defined as the fraction of features returned that are in the correct boundary (3 features for $X_1$), and recall as the fraction of the features present in the correct boundary that are returned by the algorithm. A convenient and frequently used measure that combines precision and recall is the $F_1$ measure, defined as the harmonic mean of precision and recall [18]. In Fig. 1 (top) we report 95% confidence intervals for the $F_1$ measure and execution time of $GS^{(m)}$ (margins $m = 1$ to 3) and $RGS^{(m,k)}$ (margins 1 to 3 and $k = 1000$ random subsets), using 20 data sets containing 10 to 100 variables, with the target variable $X_1$ was perturbed (inverted) by noise with 10% probability. As can be seen, the $RGS^{(m,k)}$ and $GS^{(m)}$ using the same value for margin perform comparably with respect to $F_1$, up to their 95% confidence intervals. With respect to execution time however $RGS^{(m,k)}$ exhibits much greater scalability (Fig. 1 bottom, log scale); for example, it executes in about 10 seconds on average in domains containing 100 variables, while $GS^{(m)}$ executes in 1,000 seconds on average for this domain size.

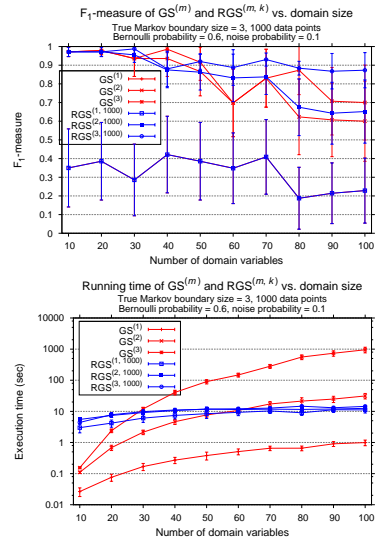

Figure 1: $GS^{(m)}$ and $RGS^{(m,k)}$ performance with respect to domain size (number of variables). **Top:** $F_1$ measure, reflecting accuracy. **Bottom:** Execution time in seconds (log scale).

We also compared $GS^{(m)}$ and $RGS^{(m,k)}$ to RELIEF [5], a well-known algorithm for feature selection that is known to be able to recover parity functions in certain cases [5]. RELIEF learns a weight for each variable and compares it to a threshold $\tau$ to decide on its inclusion in the set of relevant variables. As it has been reported [9] that RELIEF can exhibit large variance due to randomization that is necessary only for very large data sets, we instead used a deterministic variant called RELIEVED [9], whose behavior corresponds to RELIEF at the limit of infinite execution time. We calculated the $F_1$ measure for $GS^{(m)}$, $RGS^{(m,k)}$ and RELIEVED in the presence of varying amounts of noise, with noise probability ranging from 0 (no noise) to 0.4. We used domains containing 50 variables, as $GS^{(m)}$ becomes computationally demanding in larger domains. In Figure 2 (left) we show the performance of $GS^{(m)}$ and $RGS^{(m,k)}$ for $m$ equal to 1 and 3, $k = 1000$ and RELIEVED for thresholds $\tau = 0.01$ and 0.03 for various amounts of noise on the target variable. Again, each experiment was repeated 20 times to generate 95% confidence intervals. We can observe that even though $m = 1$ (equivalent to the GS algorithm) performs poorly, increasing the margin $m$ makes it more likely to recover the correct Markov boundary, and $GS^{(3)}$ ($m = 3$) recovers the exact blanket even with few (1,000) data points. RELIEVED does comparably to $GS^{(3)}$ for little noise and for a large threshold,

but appears to deteriorate for more noisy domains. As we can see it is difficult to choose the "right" threshold for RELIEVED—better performing $\tau$ at low noise can become worse in noisy environments; in particular, small $\tau$ tend to include irrelevant variables while large $\tau$ tend to miss actual members.

We also evaluated $\mathrm{GS}^{(m)}$, $\mathrm{RGS}^{(m,k)}$, and RELIEVED on benchmark and real-world data sets from the UCI Machine Learning repository. As the true Markov boundary for these is impossible to know, we used as performance measure a measure of *probabilistic isolation* by the Markov boundary returned of subsets outside the boundary. For each domain variable $X$, we measured the independence of subsets $\mathbf{Y}$ of size 1, 2 and 3 given the blanket $\mathbf{S}$ of $X$ returned by $\mathrm{GS}^{(3)}$ and RELIEVED for $\tau = 0.03$ (as this value seemed to do better in the previous set of experiments), as measured by the average p-value of the $\chi^2$ test between $X$ and $\mathbf{Y}$ given $\mathbf{S}$ (with p-values of 0 and 1 indicating ideal dependence and independence, respectively). Due to the large number of subsets outside the boundary when the boundary is small, we limited the estimation of isolation performance to 2,000 subsets per variable. We plot the results in Figure 2 (middle and right). Each point represents a variable in the corresponding data set. Points under the diagonal indicate better probabilistic isolation performance for that variable for $\mathrm{GS}^{(3)}$ compared to RELIEVED (middle plot) or to $\mathrm{RGS}^{(3,1000)}$ (right plot). To obtain a statistically significant comparison, we used the non-parametric Wilcoxon paired signed-rank test, which indicated that $\mathrm{GS}^{(3)}$ $\mathrm{RGS}^{(3,1000)}$ are statistically equivalent to each other, while both outperformed RELIEVED at the 99.99% significance level ($\alpha < 10^{-7}$).

# 7 Conclusion

In this paper we presented algorithms for the problem of feature selection in unrestricted (arbitrary distribution) domains that may contain complex interactions that only appear when the values of multiple features are considered together. We introduced two algorithms: an exact, provably correct one as well a more practical randomized anytime version, and evaluated them on on artificial, benchmark and real-world data, demonstrating that they perform well, even in the presence of noise. We also introduced the Markov Boundary Theorem that precisely characterizes the properties of a boundary, and used it to prove $m$-correctness of the exact family of algorithms presented. We made minimal assumptions that consist of only a general set of axioms that hold for every probability distribution, giving our algorithms universal applicability.

# Appendix A: Proof sketch of the Markov Boundary Theorem

*Proof sketch.* ($\Longrightarrow$ **direction**) We need to prove that if $\mathbf{S}$ is a Markov boundary of $X$ then (a) for every set $\mathbf{T} \subseteq \mathcal{U} - \mathbf{S} - \{X\}$, $(X \perp\!\!\!\perp \mathbf{T} \mid \mathbf{S} - \mathbf{T})$, and (b) for every set $\mathbf{T}' \not\subseteq \mathcal{U} - \mathbf{S}$ that does not contain $X$, $(X \not\!\perp\!\!\!\perp \mathbf{T}' \mid \mathbf{S} - \mathbf{T}')$. Case (a) is immediate from the definition of the boundary and the Decomposition theorem. Case (b) can be proven by contradiction: Assuming the independence of $\mathbf{T}'$ that contains a non-empty part $\mathbf{T}'_1$ in $\mathbf{S}$ and a part $\mathbf{T}'_2$ in $\mathcal{U} - \mathbf{S}$, we get (from Decomposition) $(X \perp\!\!\!\perp \mathbf{T}'_1 \mid \mathbf{S} - \mathbf{T}'_1)$. We can then use Contraction to show that the set $\mathbf{S} - \mathbf{T}'_1$ satisfies the independence property of a Markov boundary, i.e., that $(X \perp\!\!\!\perp \mathcal{U} - (\mathbf{S} - \mathbf{T}'_1) - \{X\} \mid \mathbf{S} - \mathbf{T}'_1)$, which contradicts the assumption that $\mathbf{S}$ is a boundary (and thus minimal).

($\Longleftarrow$ **direction**) We need to prove that if Eq. (2) holds, then $\mathbf{S}$ is a minimal Markov blanket. The proof that $\mathbf{S}$ is a blanket is immediate. We can prove minimality by contradiction: Assume $\mathbf{S} = \mathbf{S}_1 \cup \mathbf{S}_2$ with $\mathbf{S}_1$ a blanket and $\mathbf{S}_2 \neq \varnothing$ i.e., $\mathbf{S}_1$ is a blanket strictly smaller than $\mathbf{S}$. Then $(X \perp\!\!\!\perp \mathbf{S}_2 \mid \mathbf{S}_1) = (X \perp\!\!\!\perp \mathbf{S}_2 \mid \mathbf{S} - \mathbf{S}_2)$. However, since $\mathbf{S}_2 \not\subseteq \mathcal{U} - \mathbf{S}$, from Eq. (2) we get $(X \not\!\perp\!\!\!\perp \mathbf{S}_2 \mid \mathbf{S} - \mathbf{S}_2)$, which is a contradiction. $\square$

# Appendix B: Proof of $m$-Correctness of $\mathrm{GS}^{(m)}$

Let the value of the set $\mathbf{S}$ at the end of the growing phase be $\mathbf{S}_G$, its value at the end of the shrinking phase $\mathbf{S}_S$, and their difference $\mathbf{S}_\Delta = \mathbf{S}_G - \mathbf{S}_S$. The following two observations are immediate.

**Observation 1.** *For every* $\mathbf{Y} \subseteq \mathcal{U} - \mathbf{S}_G - \{X\}$ *such that* $1 \leq |\mathbf{Y}| \leq m$, $(X \perp\!\!\!\perp \mathbf{Y} \mid \mathbf{S}_G)$.

**Observation 2.** *For every* $Y \in \mathbf{S}_S$, $(X \not\!\perp\!\!\!\perp Y \mid \mathbf{S}_S - \{Y\})$.

**Lemma 2.** *Consider variables* $Y_1, Y_2, \ldots, Y_t$ *for some* $t \geq 1$ *and let* $\mathbf{Y} = \{Y_j\}_{j=1}^t$. *Assuming that Contraction holds, if* $(X \perp\!\!\!\perp Y_i \mid \mathbf{S} - \{Y_j\}_{j=1}^i)$ *for all* $i = 1, \ldots, t$, *then* $(X \perp\!\!\!\perp \mathbf{Y} \mid \mathbf{S} - \mathbf{Y})$.

*Proof.* By induction on $Y_j$, $j = 1, 2, \ldots, t$, using Contraction to decrease the conditioning set $\mathbf{S}$ down to $\mathbf{S} - \{Y_j\}_{j=1}^i$ for all $i = 1, 2, \ldots, t$. Since $\mathbf{Y} = \{Y_j\}_{j=1}^t$, we immediately obtain the desired relation $(X \perp\!\!\!\perp \mathbf{Y} \mid \mathbf{S} - \mathbf{Y})$. $\square$

Lemma 2 can be used to show that the variables found individually independent of $X$ during the shrinking phase are actually *jointly* independent of $X$, given the final set $\mathbf{S}_S$. Let $\mathbf{S}_\Delta = \{Y_1, Y_2, \ldots, Y_t\}$ be the set of variables removed (in that order) from $\mathbf{S}_G$ to form the final set $\mathbf{S}_S$ i.e., $\mathbf{S}_\Delta = \mathbf{S}_G - \mathbf{S}_S$. Using the above lemma, the following is immediate.

**Corollary 3.** *Assuming that the Contraction axiom holds, $(X \perp\!\!\!\perp \mathbf{S}_\Delta \mid \mathbf{S}_S)$.*

**Lemma 4.** *If the Contraction, Decomposition and Weak Union axioms hold, then for every set $\mathbf{T} \subseteq \mathcal{U} - \mathbf{S}_G - \{X\}$ such that $(X \perp\!\!\!\perp \mathbf{T} \mid \mathbf{S}_G)$,*

$$(X \perp\!\!\!\perp \mathbf{T} \cup (\mathbf{S}_G - \mathbf{S}_S) \mid \mathbf{S}_S). \tag{3}$$

*Furthermore $\mathbf{S}_S$ is minimal i.e., there does not exist a subset of $\mathbf{S}_S$ for which Eq. (3) is true.*

*Proof.* From Corollary 3, $(X \perp\!\!\!\perp \mathbf{S}_\Delta \mid \mathbf{S}_S)$. Also, by the hypothesis, $(X \perp\!\!\!\perp \mathbf{T} \mid \mathbf{S}_G) = (X \perp\!\!\!\perp \mathbf{T} \mid \mathbf{S}_S \cup \mathbf{S}_\Delta)$, where $\mathbf{S}_\Delta = \mathbf{S}_G - \mathbf{S}_S$ as usual. From these two relations and Contraction we obtain $(X \perp\!\!\!\perp \mathbf{T} \cup \mathbf{S}_\Delta \mid \mathbf{S}_S)$.

To prove minimality, let us assume that $\mathbf{S}_S \neq \varnothing$ (if $\mathbf{S}_S = \varnothing$ then it is already minimal). We prove by contradiction: Assume that there exists a set $\mathbf{S}' \subset \mathbf{S}_S$ such that $(X \perp\!\!\!\perp \mathbf{T} \cup (\mathbf{S}_G - \mathbf{S}') \mid \mathbf{S}')$. Let $\mathbf{W} = \mathbf{S}_S - \mathbf{S}' \neq \varnothing$. Note that $\mathbf{W}$ and $\mathbf{S}'$ are disjoint. We have that

$$\mathbf{S}_S \subseteq \mathbf{S}_S \cup \mathbf{S}_\Delta \implies \mathbf{S}_S - \mathbf{S}' \subseteq \mathbf{S}_S \cup \mathbf{S}_\Delta - \mathbf{S}' \subseteq \mathbf{T} \cup (\mathbf{S}_S \cup \mathbf{S}_\Delta - \mathbf{S}')$$
$$\implies \mathbf{W} \subseteq \mathbf{T} \cup (\mathbf{S}_S \cup \mathbf{S}_\Delta - \mathbf{S}') = \mathbf{T} \cup (\mathbf{S}_G - \mathbf{S}')$$

- Since $(X \perp\!\!\!\perp \mathbf{T} \cup (\mathbf{S}_G - \mathbf{S}') \mid \mathbf{S}')$ and $\mathbf{W} \subseteq \mathbf{T} \cup (\mathbf{S}_S \cup \mathbf{S}_\Delta - \mathbf{S}')$, from Decomposition we get $(X \perp\!\!\!\perp \mathbf{W} \mid \mathbf{S}')$.
- From $(X \perp\!\!\!\perp \mathbf{W} \mid \mathbf{S}')$ and Weak Union we have that for every $Y \in \mathbf{W}$, $(X \perp\!\!\!\perp Y \mid \mathbf{S}' \cup (\mathbf{W} - \{Y\}))$.
- Since $\mathbf{S}'$ and $\mathbf{W}$ are disjoint and since $Y \in \mathbf{W}$, $Y \notin \mathbf{S}'$. Applying the set equality $(\mathbf{A} - \mathbf{B}) \cup \mathbf{C} = (\mathbf{A} \cup \mathbf{B}) - (\mathbf{A} - \mathbf{C})$ to $\mathbf{S}' \cup (\mathbf{W} - \{Y\})$ we obtain $\mathbf{S}' \cup \mathbf{W} - (\{Y\} - \mathbf{S}') = \mathbf{S}_S - \{Y\}$.
- Therefore, $\forall Y \in \mathbf{W}, (X \perp\!\!\!\perp Y \mid \mathbf{S}_S - \{Y\})$.

However, at the end of the shrinking phase, all variables $Y$ in $\mathbf{S}_S$ (and therefore in $\mathbf{W}$, as $\mathbf{W} \subseteq \mathbf{S}_S$) have been evaluated for independence and found dependent (Observation 2). Thus, since $\mathbf{W} \neq \varnothing$, there exists at least one $Y$ such that $(X \not\perp\!\!\!\perp Y \mid \mathbf{S}_S - \{Y\})$, producing a contradiction. $\qquad \square$

**Theorem 5.** *Assuming that the Contraction, Decomposition, and Weak Union axioms hold, Algorithm 1 is $m$-correct with respect to $X$.*

*Proof.* We use the Markov Boundary Theorem. We first prove that

$$\forall \mathbf{T} \subseteq \mathcal{U} - \{X\} \text{ such that } |\mathbf{T}| \leq m, \left\{ \mathbf{T} \subseteq \mathcal{U} - \mathbf{S}_S \implies (X \perp\!\!\!\perp \mathbf{T} \mid \mathbf{S}_S - \mathbf{T}) \right\}$$

or, equivalently, $\forall \mathbf{T} \subseteq \mathcal{U} - \mathbf{S}_S - \{X\}$ such that $|\mathbf{T}| \leq m, (X \perp\!\!\!\perp \mathbf{T} \mid \mathbf{S}_S)$.

Since $\mathcal{U} - \mathbf{S}_S - \{X\} = \mathbf{S}_\Delta \cup (\mathcal{U} - \mathbf{S}_G - \{X\})$, $\mathbf{S}_\Delta$ and $\mathcal{U} - \mathbf{S}_G - \{X\}$ are disjoint, there are three kinds of sets of size $m$ or less to consider: (i) all sets $\mathbf{T} \subseteq \mathbf{S}_\Delta$, (ii) all sets $\mathbf{T} \subseteq \mathcal{U} - \mathbf{S}_G - \{X\}$, and (iii) all sets (if any) $\mathbf{T} = \mathbf{T}' \cup \mathbf{T}''$, $\mathbf{T}' \cap \mathbf{T}'' = \varnothing$, that have a non-empty part $\mathbf{T}' \subseteq \mathbf{S}_\Delta$ and a non-empty part $\mathbf{T}'' \subseteq \mathcal{U} - \mathbf{S}_G - \{X\}$.

- (i) From Corollary 3, $(X \perp\!\!\!\perp \mathbf{S}_\Delta \mid \mathbf{S}_S)$. Therefore, from Decomposition, for any set $\mathbf{T} \subseteq \mathbf{S}_\Delta$, $(X \perp\!\!\!\perp \mathbf{T} \mid \mathbf{S}_S)$.
- (ii) By Observation 1, for every set $\mathbf{T} \subseteq \mathcal{U} - \mathbf{S}_G - \{X\}$ such that $|\mathbf{T}| \leq m$, $(X \perp\!\!\!\perp \mathbf{T} \mid \mathbf{S}_G)$. By Lemma 4 we get $(X \perp\!\!\!\perp \mathbf{T} \cup \mathbf{S}_\Delta \mid \mathbf{S}_S)$, from which we obtain $(X \perp\!\!\!\perp \mathbf{T} \mid \mathbf{S}_S)$ by Decomposition.
- (iii) Since $|\mathbf{T}| \leq m$, we have that $|\mathbf{T}''| \leq m$. Since $\mathbf{T}'' \subseteq \mathcal{U} - \mathbf{S}_G - \{X\}$, by Observation 1, $(X \perp\!\!\!\perp \mathbf{T}'' \mid \mathbf{S}_G)$. Therefore, by Lemma 4, $(X \perp\!\!\!\perp \mathbf{T}'' \cup \mathbf{S}_\Delta \mid \mathbf{S}_S)$. Since $\mathbf{T}' \subseteq \mathbf{S}_\Delta \implies \mathbf{T}'' \cup \mathbf{T}' \subseteq \mathbf{T}'' \cup \mathbf{S}_\Delta$, by Decomposition to obtain $(X \perp\!\!\!\perp \mathbf{T}'' \cup \mathbf{T}' \mid \mathbf{S}_S) = (X \perp\!\!\!\perp \mathbf{T} \mid \mathbf{S}_S)$.

To complete the proof we need to prove that

$$\forall \mathbf{T} \subseteq \mathcal{U} - \{X\} \text{ such that } |\mathbf{T}| \leq m, \left\{ \mathbf{T} \not\subseteq \mathcal{U} - \mathbf{S}_S \implies (X \not\perp\!\!\!\perp \mathbf{T} \mid \mathbf{S}_S - \mathbf{T}) \right\}.$$

Let $\mathbf{T} = \mathbf{T}_1 \cup \mathbf{T}_2$, with $\mathbf{T}_1 \subseteq \mathbf{S}_S$ and $\mathbf{T}_2 \subseteq \mathcal{U} - \mathbf{S}_S$. Since $\mathbf{T} \not\subseteq \mathcal{U} - \mathbf{S}_S$, $\mathbf{T}_1$ contains at least one variable $Y \in \mathbf{S}_S$. From Observation 2, $(X \not\perp\!\!\!\perp Y \mid \mathbf{S}_S - \{Y\})$. From this and (the contrapositive of) Weak Union, we get $(X \not\perp\!\!\!\perp \{Y\} \cup (\mathbf{T}_1 - \{Y\}) \mid \mathbf{S}_S - \{Y\} - (\mathbf{T}_1 - \{Y\})) = (X \not\perp\!\!\!\perp \mathbf{T}_1 \mid \mathbf{S}_S - \mathbf{T}_1)$. From (the contrapositive of) Decomposition we get $(X \not\perp\!\!\!\perp \mathbf{T}_1 \cup \mathbf{T}_2 \mid \mathbf{S}_S - \mathbf{T}_1) = (X \not\perp\!\!\!\perp \mathbf{T} \mid \mathbf{S}_S - \mathbf{T}_1)$, which is equal to $(X \not\perp\!\!\!\perp \mathbf{T} \mid \mathbf{S}_S - \mathbf{T}_1 - \mathbf{T}_2) = (X \not\perp\!\!\!\perp \mathbf{T} \mid \mathbf{S}_S - \mathbf{T})$ as $\mathbf{S}_S$ and $\mathbf{T}_2$ are disjoint. $\qquad \square$

# References

[1] Isabelle Guyon and André Elisseeff. An introduction to variable and feature selection. *Journal of Machine Learning Research*, 3:1157–1182, 2003.

[2] Daphne Koller and Mehran Sahami. Toward optimal feature selection. In *Proceedings of the Tenth International Conference on Machine Learning (ICML)*, pages 284–292, 1996.

[3] P. M. Narendra and K. Fukunaga. A branch and bound algorithm for feature subset selection. *IEEE Transactions on Computers*, C-26(9):917–922, 1977.

[4] H. Almuallim and T. G. Dietterich. Learning with many irrelevant features. In *Proceedings of the National Conference on the Americal Association for Artifical Intelligence (AAAI)*, 1991.

[5] K. Kira and L. A. Rendell. The feature selection problem: Traditional methods and a new algorithm. In *Proceedings of the National Conference on the Americal Association for Artifical Intelligence (AAAI)*, pages 129–134, 1992.

[6] M. Dash and H. Liu. Feature selection for classification. *Intelligent Data Analysis*, 1(3): 131–156, 1997.

[7] Huan Liu and Hiroshi Motoda, editors. *Feature Extraction, Construction and Selection: A Data Mining Perspective*, volume 453 of *The Springer International Series in Engineering and Computer Science*. 1998.

[8] Avrim Blum and Pat Langley. Selection of relevant features and examples in machine learning. *Artificial Intelligence*, 97(1-2):245–271, 1997.

[9] R. Kohavi and G. H. John. Wrappers for feature subset selection. *Artificial Intelligence*, 97 (1-2):273–324, 1997.

[10] Dimitris Margaritis and Sebastian Thrun. Bayesian network induction via local neighborhoods. In *Advances in Neural Information Processing Systems 12 (NIPS)*, 2000.

[11] I. Tsamardinos, C. Aliferis, and A. Statnikov. Algorithms for large scale Markov blanket discovery. In *Proceedings of the 16th International FLAIRS Conference*, 2003.

[12] I. Tsamardinos, C. Aliferis, and A. Statnikov. Time and sample efficient discovery of Markov blankets and direct causal relations. In *Proceedings of the 9th ACM SIGKDD International Conference on Knowledge Discovery and Data Mining*, pages 673–678, 2003.

[13] N. Meinshausen and P. Bühlmann. High-dimensional graphs and variable selection with the Lasso. *Annals of Statistics*, 34:1436–1462, 2006.

[14] Judea Pearl. *Probabilistic Reasoning in Intelligent Systems: Networks of Plausible Inference*. 1988.

[15] Michael Kearns and Umesh V. Vazirani. *An Introduction to Computational Learning Theory*. MIT Press, 1994.

[16] A. Agresti. *Categorical Data Analysis*. John Wiley and Sons, 1990.

[17] M. Kearns. Efficient noise-tolerant learning from statistical queries. *J. ACM*, 45(6):983–1006, 1998.

[18] C. J. van Rijsbergen. *Information Retrieval*. Butterworth-Heinemann, London, 1979.

